# Adaptive Elastic Models for Hand-Printed Character Recognition

**Geoffrey E. Hinton, Christopher K. I. Williams and Michael D. Revow**
Department of Computer Science, University of Toronto
Toronto, Ontario, Canada M5S 1A4

## Abstract

Hand-printed digits can be modeled as splines that are governed by about 8 control points. For each known digit, the control points have preferred "home" locations, and deformations of the digit are generated by moving the control points away from their home locations. Images of digits can be produced by placing Gaussian ink generators uniformly along the spline. Real images can be recognized by finding the digit model most likely to have generated the data. For each digit model we use an elastic matching algorithm to minimize an energy function that includes both the deformation energy of the digit model and the log probability that the model would generate the inked pixels in the image. The model with the lowest total energy wins. If a uniform noise process is included in the model of image generation, some of the inked pixels can be rejected as noise as a digit model is fitting a poorly segmented image. The digit models learn by modifying the home locations of the control points.

## 1 Introduction

Given good bottom-up segmentation and normalization, feedforward neural networks are an efficient way to recognize digits in zip codes. (le Cun et al., 1990). However, in some cases, it is not possible to correctly segment and normalize the digits without using knowledge of their shapes, so to achieve close to human performance on images of whole zip codes it will be necessary to use models of shapes to influence the segmentation and normalization of the digits. One way of doing this is to use a large cooperative network that simultaneously segments, normalizes and recognizes all of the digits in a zip code. A first step in this direction is to take a poorly segmented image of a single digit and to explain the image properly in terms of an appropriately normalized, deformed digit model plus noise. The ability of the model to reject some parts of the image as noise is the first step towards model-driven segmentation.

## 2   Elastic models

One technique for recognizing a digit is to perform an elastic match with many different exemplars of each known digit-class and to pick the class of the nearest neighbor. Unfortunately this requires a large number of elastic matches, each of which is expensive. By using one elastic model to capture all the variations of a given digit we greatly reduce the number of elastic matches required. Burr (1981a, 1981b) has investigated several types of elastic model and elastic matching procedure. We describe a different kind of elastic model that is based on splines. Each elastic model contains parameters that define an ideal shape and also define a deformation energy for departures from this ideal. These parameters are initially set by hand but can be improved by learning. They are an efficient way to represent the many possible instances of a given digit.

Each digit is modelled by a deformable spline whose shape is determined by the positions of 8 control points. Every point on the spline is a weighted average of four control points, with the weighting coefficients changing smoothly as we move along the spline. [1] To generate an ideal example of a digit we put the 8 control points at their home locations for that model. To deform the digit we move the control points away from their home locations. Currently we assume that, for each model, the control points have independent, radial Gaussian distributions about their home locations. So the negative log probability of a deformation (its energy) is proportional to the sum of the squares of the departures of the control points from their home locations.

The deformation energy function only penalizes shape *deformations*. Translation, rotation, dilation, elongation, and shear do not change the shape of an object so we want the deformation energy to be invariant under these affine transformations. We achieve this by giving each model its own "object-based frame". Its deformation energy is computed relative to this frame, not in image coordinates. When we fit the model to data, we repeatedly recompute the best affine transformation between the object-based frame and the image (see section 4). The repeated recomputation of the affine transform during the model fit means that the shape of the digit is influencing the normalization.

Although we will use our digit models for recognizing images, it helps to start by considering how we would use them for generating images. The generative model is an elaboration of the probabilistic interpretation of the elastic net given by Durbin, Szeliski & Yuille (1989). Given a particular spline, we space a number of "beads" uniformly along the spline. Each bead defines the center of a Gaussian ink generator. The number of beads on the spline and the variance of the ink generators can easily be changed without changing the spline itself.

To generate a noisy image of a particular digit class, run the following procedure:

- Pick an affine transformation from the model's intrinsic reference frame to the image frame (i.e. pick a size, position, orientation, slant and elongation for the digit).

- Pick a deformation of the model (i.e. move the control points away from their home locations). The probability of picking a deformation is $\frac{1}{Z} e^{-E_{deform}}$
- Repeat many times:
  **Either** (with probability $\pi_{noise}$) add a randomly positioned noise pixel
  **Or** pick a bead at random and generate a pixel from the Gaussian distribution defined by the bead.

## 3  Recognizing isolated digits

We recognize an image by finding which model is most likely to have generated it. Each possible model is fitted to the image and the one that has the lowest cost fit is the winner. The cost of a fit is the negative log probability of generating the image given the model.

$$E_{ideal} = -log \int\limits_{\substack{I \in model \\ instances}} P(I)\, P(image \mid I)\, dI \qquad (1)$$

We can approximate this by just considering the best fitting model instance [2] and ignoring the fact that the model should not generate ink where there is no ink in the image:[3]

$$E = \lambda\, E_{deform} - \sum_{\substack{inked \\ pixels}} \log P(pixel \mid best\ model\ instance) \qquad (2)$$

The probability of an inked pixel is the sum of the probabilities of all the possible ways of generating it from the mixture of Gaussian beads or the uniform noise field.

$$P(i) = \frac{\pi_{noise}}{N} + \frac{\pi_{model}}{B} \sum_{beads} P_b(i) \qquad (3)$$

where $N$ is the total number of pixels, $B$ is the number of beads, $\pi$ is a mixing proportion, and $P_b(i)$ is the probability density of pixel $i$ under Gaussian bead $b$.

## 4  The search procedure for fitting a model to an image

Every Gaussian bead in a model has the same variance. When fitting data, we start with a big variance and gradually reduce it as in the elastic net algorithm of Durbin and Willshaw (1987) . Each iteration of the elastic matching algorithm involves three steps:

- Given the current locations of the Gaussians, compute the responsibility that each Gaussian has for each inked pixel. This is just the probability of generating the pixel from that Gaussian, normalized by the total probability of generating the pixel.

- Assuming that the responsibilities remain fixed, as in the EM algorithm of Dempster, Laird and Rubin (1977), we invert a $16 \times 16$ matrix to find the image locations for the 8 control points at which the forces pulling the control points towards their home locations are balanced by the forces exerted on the control points by the inked pixels. These forces come via the forces that the inked pixels exert on the Gaussian beads.

- Given the new image locations of the control points, we recompute the affine transformation from the object-based frame to the image frame. We choose the affine transformation that minimizes the sum of the squared distances, in object-based coordinates, between the control points and their home locations. The residual squared differences determine the deformation energy.

Some stages in the fitting of a model to data are shown in Fig. 1. This search technique avoids nearly all local minima when fitting models to isolated digits. But if we get a high deformation energy in the best fitting model, we can try alternative starting configurations for the models.

## 5   Learning the digit models

We can do discriminative learning by adjusting the home positions and variances of the control points to minimize the objective function

$$C = - \sum_{\substack{training \\ cases}} \log p(correct \ digit), \quad p(correct \ digit) = \frac{e^{-E_{correct}}}{\sum_{all \ digits} e^{-E_{digit}}} \quad (4)$$

For a model parameter such as the $x$ coordinate of the home location of one of the control points we need $\partial C / \partial x$ in order to do gradient descent learning. Equation 4 allows us to express $\partial C / \partial x$ in terms of $\partial E / \partial x$ but there is a subtle problem: Changing a parameter of an elastic model causes a simple change in the energy of the configuration that the model previously settled to, but the model no longer settles to that configuration.   So it appears that we need to consider how the energy is affected by the change in the configuration. Fortunately, derivatives are simple at an energy minimum because small changes in the configuration make no change in the energy (to first order). Thus the inner loop settling leads to simple derivatives for the outer loop learning, as in the Boltzmann machine (Hinton, 1989).

## 6   Results on the hand-filtered dataset

We are trying out the scheme out on a relatively simple task - we have a model of a two and a model of a three, and we want the two model to win on "two" images, and the three model to win on "three" images.

We have tried many variations of the character models, the preprocessing, the initial affine transformations of the models, the annealing schedule for the variances, the

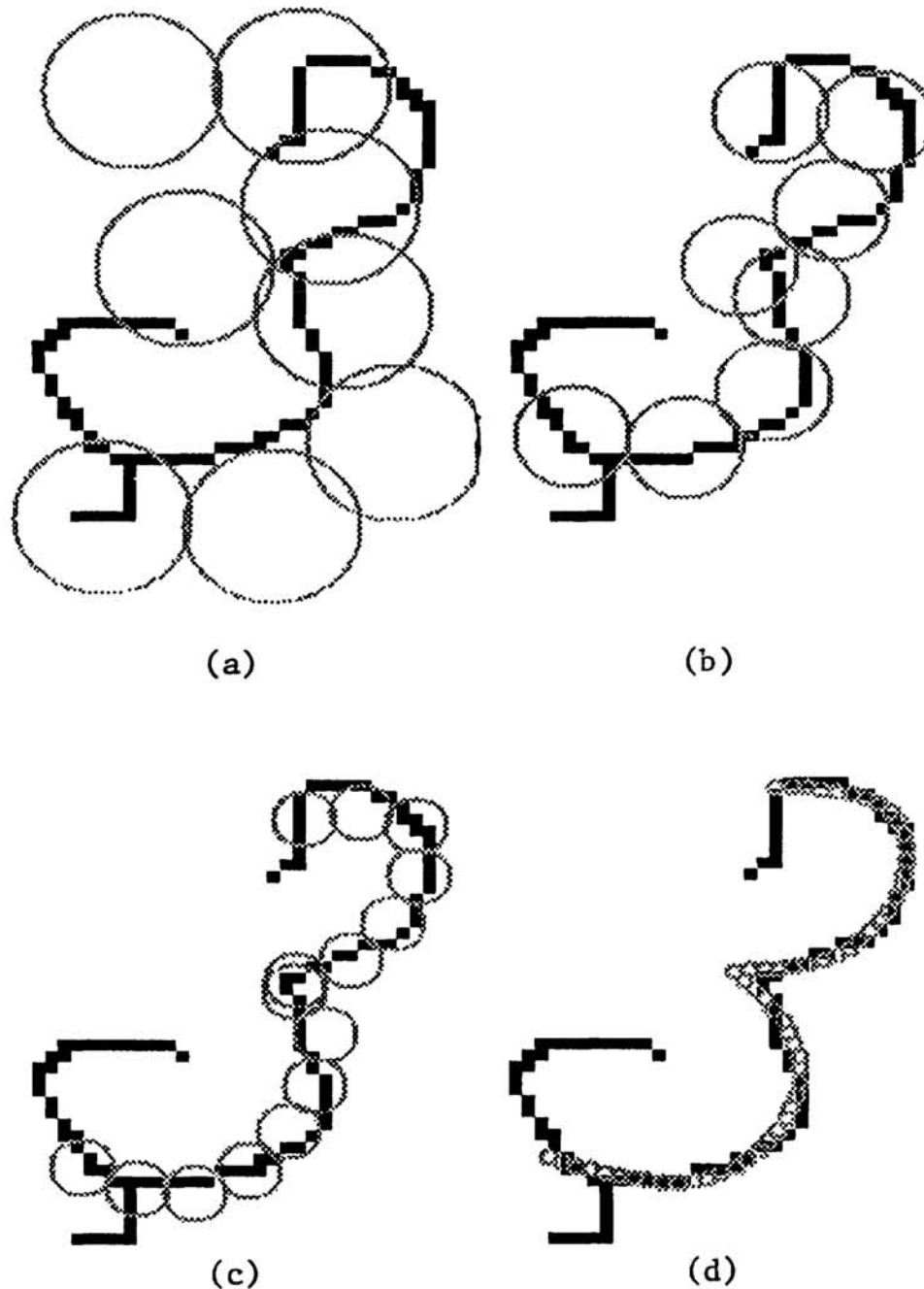

(a)                                    (b)

(c)                                    (d)

Figure 1: The sequence (a) to (d) shows some stages of fitting a model 3 to some data. The grey circles represent the beads on the spline, and the radius of the circle represents the standard deviation of the Gaussian. (a) shows the initial configuration, with eight beads equally spaced along the spline. In (b) and (c) the variance is progressively decreased and the number of beads is increased. The final fit using 60 beads is shown in (d). We use about three iterations at each of five variances on our "annealing schedule". In this example, we used $\pi_{noise} = 0.3$ which makes it cheaper to explain the extraneous noise pixels and the flourishes on the ends of the 3 as noise rather than deforming the model to bring Gaussian beads close to these pixels.

mixing proportion of the noise, and the relative importance of deformation energy versus data-fit energy.

Our current best performance is 10 errors (1.6%) on a test set of 304 two's and 304 three's. We reject cases if the best-fitting model is highly deformed, but on this test set the deformation energy never reached the rejection criterion. The training set has 418 cases, and we have a validation set of 200 cases to tell us when to stop learning. Figure 2 shows the effect of learning on the models. The initial affine transform is defined by the minimal vertical rectangle around the data.

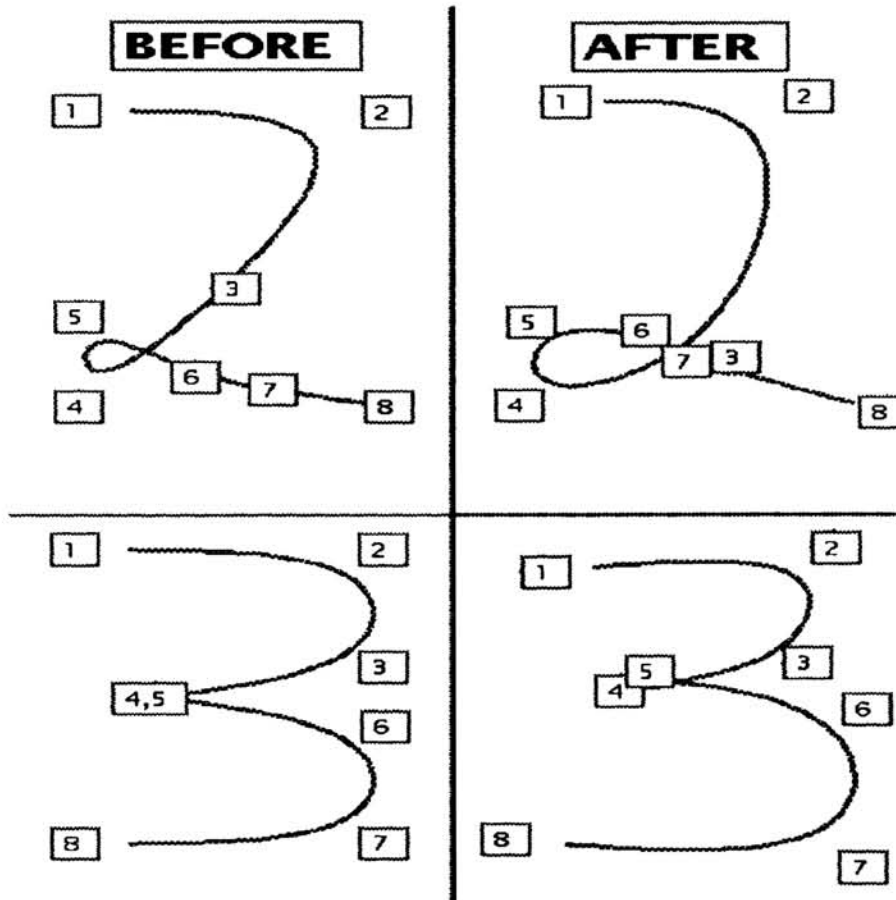

Figure 2: The two and three models before and after learning. The control points are labelled 1 through 8. We used maximum likelihood learning in which each digit model is trained only on instances of that digit. After each pass through all those instances, the home location of each control point (in the object-based frame) is redefined to be the average location of the control point in the final fits of the model of the digit to the instances of the digit. Most of the improvement in performance occurred after the fist pass, and after five updates of the home locations of the control points, performance on the validation set started to decrease. Similar results were obtained with discriminative training. We could also update the variance of each control point to be its variance in the final fits, though we did not adapt the variances in this simulation.

The images are preprocessed to eliminate variations due to stroke-width and paper and ink intensities. First, we use a standard local thresholding algorithm to make a binary decision for each pixel. Then we pick out the five largest connected components (hopefully digits). We put a box around each component, then thin all the data in the box. If we ourselves cannot recognize the resulting image we eliminate it from the data set. The training, validation and test data is all from the training portion of the United States Postal Service Handwritten ZIP Code Database (1987) which was made available by the USPS Office of Advanced Technology.

## 7    Discussion

Before we tried using splines to model digits, we used models that consisted of a fixed number of Gaussian beads with elastic energy constraints operating between neighboring beads. To constrain the curvature we used energy terms that involved triples of beads. With this type of energy function, we had great difficulty using a single model to capture topologically different instances of a digit. For example, when the central loop of a 3 changes to a cusp and then to an open bend, the sign of the curvature reverses. With a spline model it is easy to model these topological variants by small changes in the relative vertical locations of the central two control points (see figure 2). This advantage of spline models is pointed out by (Edelman, Ullman and Flash, 1990) who use a different kind of spline that they fit to character data by directly locating candidate knot points in the image.

Spline models also make it easy to increase the number of Gaussian beads as their variance is decreased. This coarse-to-fine strategy is much more efficient than using a large number of beads at all variances, but it is much harder to implement if the deformation energy explicitly depends on particular bead locations, since changing the number of beads then requires a new function for the deformation energy.

In determining where on the spline to place the Gaussian beads, we initially used a fixed set of blending coefficients for each bead. These coefficients are the weights used to specify the bead location as a weighted center of gravity of the locations of 4 control points. Unfortunately this yields too few beads in portions of a digit such as a long tail of a 2 which are governed by just a few control points. Performance was much improved by spacing the beads uniformly along the curve.

By using spline models, we build in a lot of prior knowledge about what characters look like, so we can describe the shape of a character using only a small number of parameters (16 coordinates and 8 variances). This means that the learning is exploring a much smaller space than a conventional feed-forward network. Also, because the parameters are easy to interpret, we can start with fairly good initial models of the characters. So learning only requires a few updates of the parameters.

Obvious extensions of the deformation energy function include using elliptical Gaussians for the distributions of the control points, or using full covariance matrices for neighboring pairs of control points. Another obvious modification is to use elliptical rather than circular Gaussians for the beads. If strokes curve gently relative to their thickness, the distribution of ink can be modelled much better using elliptical Gaussians. However, an ellipse takes about twice as many operations to fit and is not helpful in regions of sharp curvature. Our simulations suggest that, on average, two circular beads are more flexible than one elliptical bead.

Currently we do not impose any penalty on extremely sheared or elongated affine transformations, though this would probably improve performance. Having an explicit representation of the affine transformation of each digit should prove very helpful for recognizing multiple digits, since it will allow us to impose a penalty on differences in the affine transformations of neighboring digits.

Presegmented images of single digits contain many different kinds of noise that cannot be eliminated by simple bottom-up operations. These include descenders, underlines, and bits of other digits; corrections; dirt in recycled paper; smudges and misplaced postal franks. To really understand the image we probably need to model a wide variety of structured noise. We are currently experimenting with one simple way of incorporating noise models. After each digit model has been used to segment a noisy image into one digit instance plus noise, we try to fit more complicated noise models to the residual noise. A good fit greatly decreases the cost of that noise and hence improves this interpretation of the image. We intend to handle flourishes on the ends of characters in this way rather than using more elaborate digit models that include optional flourishes.

One of our main motivations in developing elastic models is the belief that a strong prior model should make learning easier, should reduce confident errors, and should allow top-down segmentation. Although we have shown that elastic spline models can be quite effective, we have not yet demonstrated that they are superior to feedforward nets and there is a serious weakness of our approach: Elastic matching is slow. Fitting the models to the data takes *much* more computation than a feedforward net. So in the same number of cycles, a feedforward net can try many alternative bottom-up segmentations and normalizations and select the overall segmentation that leads to the most recognizable digit string.

### Acknowledgements
This research was funded by Apple and by the Ontario Information Technology Research Centre. We thank Allan Jepson and Richard Durbin for suggesting spline models.

## Footnotes

[1] In computing the weighting coefficients we use a cubic B-spline and we treat the first and last control points as if they were doubled.

[2] In effect, we are assuming that the integral in equation 1 can be approximated by the height of the highest peak, and so we are ignoring variations between models in the width of the peak or the number of peaks.

[3] If the inked pixels are rare, poor models sin mainly by not inking those pixels that should be inked rather than by inking those pixels that should not be inked.

### References

Burr, D. J. (1981a). A dynamic model for image registration. *Comput. Graphics Image Process.*, 15:102–112.

Burr, D. J. (1981b). Elastic matching of line drawings. *IEEE Trans. Pattern Analysis and Machine Intelligence*, 3(6):708–713.

Dempster, A. P., Laird, N. M., and Rubin, D. B. (1977). Maximum likelihood from incomplete data via the EM algorithm. *Proc. Roy. Stat. Soc.*, B-39:1–38.

Durbin, R., Szeliski, R., and Yuille, A. L. (1989). An analysis of the elastic net approach to the travelling salesman problem. *Neural Computation*, 1:348–358.

Durbin, R. and Willshaw, D. (1987). An analogue approach to the travelling salesman problem. *Nature*, 326:689–691.

Edelman, S., Ullman, S., and Flash, T. (1990). Reading cursive handwriting by alignment of letter prototypes. *Internat. Journal of Comput. Vision*, 5(3):303–331.

Hinton, G. E. (1989). Deterministic Boltzmann learning performs steepest descent in weight-space. *Neural Computation*, 1:143–150.

le Cun, Y., Boser, B., Denker, J., Henderson, D., Howard, R., Hubbard, W., and Jackel, L. (1990). Handwritten digit recognition with a back-propagation network. In *Advances in Neural Information Processing Systems 2*, pages 396–404. Morgan Kaufmann.


